# TrueSkill$^\text{TM}$: A Bayesian Skill Rating System

**Ralf Herbrich**
Microsoft Research Ltd.
Cambridge, UK
*rherb@microsoft.com*

**Tom Minka**
Microsoft Research Ltd.
Cambridge, UK
*minka@microsoft.com*

**Thore Graepel**
Microsoft Research Ltd.
Cambridge, UK
*thoreg@microsoft.com*

## Abstract

We present a new Bayesian skill rating system which can be viewed as a generalisation of the Elo system used in Chess. The new system tracks the uncertainty about player skills, explicitly models draws, can deal with any number of competing entities and can infer individual skills from team results. Inference is performed by approximate message passing on a factor graph representation of the model. We present experimental evidence on the increased accuracy and convergence speed of the system compared to Elo and report on our experience with the new rating system running in a large-scale commercial online gaming service under the name of *TrueSkill*.

## 1  Introduction

Skill ratings in competitive games and sports serve three main functions. First, they allow players to be matched with other players of similar skill leading to interesting, balanced matches. Second, the ratings can be made available to the players and to the interested public and thus stimulate interest and competition. Thirdly, ratings can be used as criteria of qualification for tournaments. With the advent of online gaming, the interest in rating systems has increased dramatically because the quality of the online experience of millions of players each day are at stake.

In 1959, Arpad Elo developed a statistical rating system for Chess, which was adopted by the World Chess Federation FIDE in 1970 [4]. The key idea behind the *Elo system* [2] is to model the probability of the possible game outcomes as a function of the two players' skill ratings $s_1$ and $s_2$. In a game each player $i$ exhibits performance $p_i \sim \mathcal{N}(p_i; s_i, \beta^2)$ normally distributed around their skills $s_i$ with fixed variance $\beta^2$. The probability that player 1 wins is given by the probability that his performance $p_1$ exceeds the opponent's performance $p_2$,

$$P(p_1 > p_2 | s_1, s_2) = \Phi\left(\frac{s_1 - s_2}{\sqrt{2}\beta}\right) , \tag{1}$$

where $\Phi$ denotes the cumulative density of a zero-mean unit-variance Gaussian. After the game, the skill ratings $s_1$ and $s_2$ are updated such that the observed game outcome becomes more likely and $s_1 + s_2 = \text{const.}$ is maintained. Let $y = +1$ if player 1 wins, $y = -1$ if player 2 wins and $y = 0$ if a draw occurs. Then the resulting (linearised) Elo update is given by $s_1 \leftarrow s_1 + y\Delta$, $s_2 \leftarrow s_2 - y\Delta$ and

$$\Delta = \underbrace{\alpha\beta\sqrt{\pi}}_{K-\text{Factor}} \left(\frac{y+1}{2} - \Phi\left(\frac{s_1 - s_2}{\sqrt{2}\beta}\right)\right) ,$$

where $0 < \alpha < 1$ determines the weighting of the new evidence versus the old estimate. Most currently used Elo variants use a logistic distribution instead of a Gaussian because it is argued to provide a better fit for Chess data. From the point of view of statistics the

Elo system addresses the problem of estimating from paired comparison data [1] with the Gaussian variant corresponding to the *Thurstone Case V* model and the logistic variant to the *Bradley-Terry* model.

In the Elo system, a player's rating is regarded as provisional as long as it is based on less than a fixed number of, say, 20 games. This problem was addressed by Mark Glickman's Bayesian rating system *Glicko* [5] which introduces the idea of modeling the belief about a player's skill as a Gaussian belief distribution characterised by a mean $\mu$ and a variance $\sigma^2$.

An important new application of skill rating systems are multiplayer online games that greatly benefit from the ability to create online matches in which the participating players have roughly even skills and hence enjoyable, fair and exciting game experiences. Multiplayer online games provide the following challenges:

1. Game outcomes often refer to teams of players yet a skill rating for individual players is needed for future matchmaking.

2. More than two players or teams compete such that the game outcome is a permutation of teams or players rather than just a winner and a loser.

In this paper we present a new rating system, *TrueSkill*, that addresses both these challenges in a principled Bayesian framework. We express the model as a factor graph (Section 2) and use approximate message passing (Section 3) to infer the marginal belief distribution over the skill of each player. In Section 4 we present experimental results on real-world data generated by Bungie Studios during the beta testing of the Xbox title Halo 2 and we report on our experience with the rating system running in the Xbox Live service.

## 2    Factor Graphs for Ranking

From among a population of $n$ players $\{1, \ldots, n\}$ in a game let $k$ teams compete in a match. The team assignments are specified by $k$ non-overlapping subsets $A_j \subset \{1, \ldots, n\}$ of the player population, $A_i \cap A_j = \emptyset$ if $i \neq j$. The outcome $\mathbf{r} := (r_1, \ldots, r_k) \in \{1, \ldots, k\}$ is specified by a rank $r_j$ for each team $j$, with $r = 1$ indicating the winner and with the possibility of draws when $r_i = r_j$. The ranks are derived from the scoring rules of the game.

We model the probability $P(\mathbf{r}|\mathbf{s}, A)$ of the game outcome $\mathbf{r}$ given the skills $\mathbf{s}$ of the participating players and the team assignments $A := \{A_1, \ldots, A_k\}$. From Bayes' rule we obtain the posterior distribution

$$p(\mathbf{s}|\mathbf{r}, A) = \frac{P(\mathbf{r}|\mathbf{s}, A) \, p(\mathbf{s})}{P(\mathbf{r}|A)} . \tag{2}$$

We assume a factorising Gaussian prior distribution, $p(\mathbf{s}) := \prod_{i=1}^{n} \mathcal{N}(s_i; \mu_i, \sigma_i^2)$. Each player $i$ is assumed to exhibit a performance $p_i \sim \mathcal{N}(p_i; s_i, \beta^2)$ in the game, centred around their skill $s_i$ with fixed variance $\beta^2$. The performance $t_j$ of team $j$ is modeled as the sum of the performances of its members, $t_j := \sum_{i \in A_j} p_i$. Let us reorder the teams in ascending order of rank, $r_{(1)} \leq r_{(2)} \leq \cdots \leq r_{(k)}$. Disregarding draws, the probability of a game outcome $\mathbf{r}$ is modeled as

$$P(\mathbf{r}|\{t_1, \ldots, t_k\}) = P\left(t_{r_{(1)}} > t_{r_{(2)}} > \cdots > t_{r_{(k)}}\right) ,$$

that is, the order of performances generates the order in the game outcome. If draws are permitted the winning outcome $r_{(j)} < r_{(j+1)}$ requires $t_{r_{(j)}} > t_{r_{(j+1)}} + \varepsilon$ and the draw outcome $r_{(j)} = r_{(j+1)}$ requires $|t_{r_{(j)}} - t_{r_{(j+1)}}| \leq \varepsilon$, where $\varepsilon > 0$ is a draw margin that can be calculated from the assumed probability of draw.[1]

We need to be able to report skill estimates after each game and will therefore use an online learning scheme referred to as *Gaussian density filtering* [8]. The posterior distribution is approximated to be Gaussian and is used as the prior distribution for the next game. If the

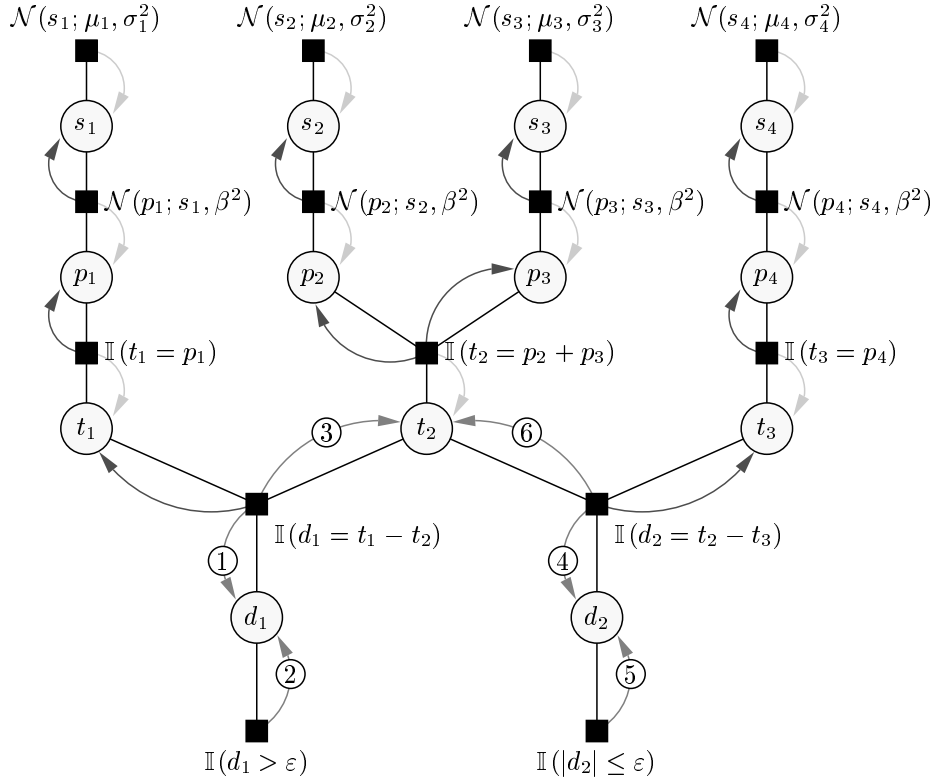

Figure 1: An example TrueSkill factor graph. There are four types of variables: $s_i$ for the *skills* of all players, $p_i$ for the *performances* of all players, $t_i$ for the *performances* of all *teams* and $d_j$ for the *team performance differences*. The first row of factors encode the (product) prior; the product of the remaining factors characterises the likelihood for the game outcome Team 1 > Team 2 = Team 3. The arrows indicate the optimal message passing schedule: First, all light arrow messages are updated from top to bottom. In the following, the schedule over the team performance (difference) nodes are iterated in the order of the numbers. Finally, the posterior over the skills is computed by updating all the dark arrow messages from bottom to top.

skills are expected to vary over time, a Gaussian dynamics factor $\mathcal{N}(s_{i,t+1}; s_{i,t}, \gamma^2)$ can be introduced which leads to an additive variance component of $\gamma^2$ in the subsequent prior.

Let us consider a game with $k = 3$ teams with team assignments $A_1 = \{1\}$, $A_2 = \{2, 3\}$ and $A_3 = \{4\}$. Let us further assume that team 1 is the winner and that teams 2 and 3 draw, i.e., $\mathbf{r} := (1, 2, 2)$. We can represent the resulting joint distribution $p(\mathbf{s}, \mathbf{p}, \mathbf{t} | \mathbf{r}, A)$ by the factor graph depicted in Figure 1.

A factor graph is a bi-partite graph consisting of variable and factor nodes, shown in Figure 1 as gray circles and black squares, respectively. The function represented by a factor graph—in our case the joint distribution $p(\mathbf{s}, \mathbf{p}, \mathbf{t} | \mathbf{r}, A)$—is given by the product of all the (potential) functions associated with each factor. The structure of the factor graph gives information about the dependencies of the factors involved and is the basis of efficient inference algorithms. Returning to Bayes rule (2), the quantities of interest are the posterior distribution $p(s_i | \mathbf{r}, A)$ over skills given game outcome $\mathbf{r}$ and team associations $A$. The $p(s_i | \mathbf{r}, A)$ are calculated from the joint distribution integrating out the individual performances $\{p_i\}$ and the team performances $\{t_i\}$,

$$p(\mathbf{s} | \mathbf{r}, A) = \int_{-\infty}^{\infty} \cdots \int_{-\infty}^{\infty} p(\mathbf{s}, \mathbf{p}, \mathbf{t} | \mathbf{r}, A) \, d\mathbf{p} \, d\mathbf{t}.$$

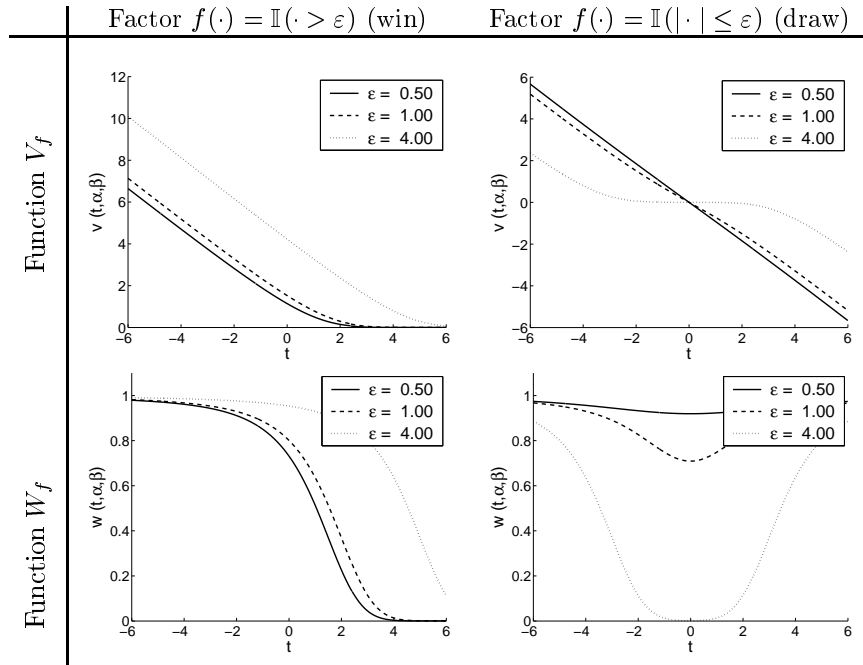

| Factor $f(\cdot) = \mathbb{I}(\cdot > \varepsilon)$ (win) | Factor $f(\cdot) = \mathbb{I}(|\cdot| \le \varepsilon)$ (draw) |

Figure 2: Update rules for the the approximate marginals for different values of the draw margin $\varepsilon$. For a two-team game, the parameter $t$ represents the difference of team performances between winner and loser. Hence, in the "win" column (left) negative values of $t$ indicate a surprise outcome leading to a large update. In the "draw" column (right) any stark deviation of team performances is surprising and leads to a large update.

# 3  Approximate Message Passing

The sum-product algorithm in its formulation for factor graphs [7] exploits the sparse connection structure of the graph to perform efficient inference of single-variable marginals by message passing. The message passing for continuous variables is characterised by the following equations (these follow directly from the distributive law):

$$p(v_k) = \prod_{f \in F_{v_k}} m_{f \to v_k}(v_k) \tag{3}$$

$$m_{f \to v_j(v_j)} = \int \cdots \int f(\mathbf{v}) \prod_{i \ne j} m_{v_i \to f}(v_i) \, d\mathbf{v}_{\setminus j} \tag{4}$$

$$m_{v_k \to f}(v_k) = \prod_{\bar{f} \in F_{v_k} \setminus \{f\}} m_{\bar{f} \to v_k}(v_k) , \tag{5}$$

where $F_{v_k}$ denotes the set of factors connected to variable $v_k$ and $\mathbf{v}_{\setminus j}$ denotes the components of the vector $\mathbf{v}$ except for its $j$th component. If the factor graph is acyclic and the messages can be calculated and represented exactly then each message needs to be calculated only once and the marginals $p(v_k)$ can be calculated from the messages by virtue of (3).

As can be seen from Figure 1 the TrueSkill factor graph is in fact acyclic and the majority of messages can be represented compactly as 1–dimensional Gaussians. However, (4) shows that messages 2 and 5 from the comparison factors $\mathbb{I}(\cdot > \varepsilon)$ or $\mathbb{I}(|\cdot| \le \varepsilon)$ to the performance differences $d_i$ in Figure 1 are non Gaussian—in fact, the true message would be the (non-Gaussian) factor itself.

Following the *Expectation Propagation* algorithm [8], we approximate these messages as well as possible by approximating the marginal $p(d_i)$ via moment matching resulting in a Gaussian $\hat{p}(d_i)$ with the same mean and variance as $p(d_i)$. For Gaussian distributions,

| Factor | Update equation |
|---|---|
| 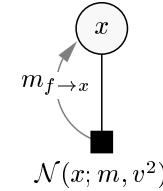 $\mathcal{N}(x; m, v^2)$ | $\pi_x^{\text{new}} \leftarrow \pi_x + \dfrac{1}{v^2}$ <br><br> $\tau_x^{\text{new}} \leftarrow \tau_x + \dfrac{m}{v^2}$ |
| 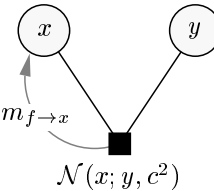 $\mathcal{N}(x; y, c^2)$ | $\pi_{f \to x}^{\text{new}} \leftarrow a\left(\pi_y - \pi_{f \to y}\right)$ <br> $\tau_{f \to x}^{\text{new}} \leftarrow a\left(\tau_y - \tau_{f \to y}\right)$ <br><br> $a := \left(1 + c^2\left(\pi_y - \pi_{f \to y}\right)\right)^{-1}$ <br> $m_{f \to y}$ follows from $\mathcal{N}\left(x; y, c^2\right) = \mathcal{N}\left(y; x, c^2\right)$. |
| 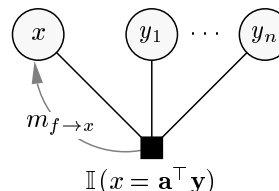 $\mathbb{I}(x = \mathbf{a}^\top \mathbf{y})$ | $\pi_{f \to x}^{\text{new}} \leftarrow \left(\displaystyle\sum_{j=1}^{n} \frac{a_j^2}{\pi_{y_j} - \pi_{f \to y_j}}\right)^{-1}$ <br><br> $\tau_{f \to x}^{\text{new}} \leftarrow \pi_{f \to x}^{\text{new}} \cdot \left(\displaystyle\sum_{j=1}^{n} a_j \cdot \frac{\tau_{y_j} - \tau_{f \to y_j}}{\pi_{y_j} - \pi_{f \to y_j}}\right)$ |
| 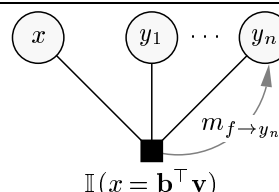 $\mathbb{I}(x = \mathbf{b}^\top \mathbf{y})$ | 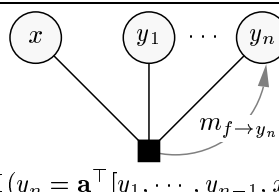 $\mathbb{I}\left(y_n = \mathbf{a}^\top [y_1, \cdots, y_{n-1}, x]\right)$ $\quad \mathbf{a} = \dfrac{1}{b_n} \cdot \begin{bmatrix} -b_1 \\ \vdots \\ -b_{n-1} \\ +1 \end{bmatrix}$ |
| 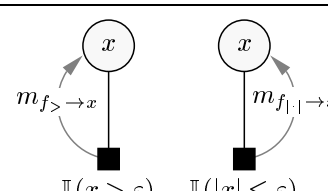 $\mathbb{I}(x > \varepsilon) \quad \mathbb{I}(|x| \leq \varepsilon)$ | $\pi_x^{\text{new}} \leftarrow \dfrac{c}{1 - W_f\left(d/\sqrt{c}, \varepsilon\sqrt{c}\right)}$ <br><br> $\tau_x^{\text{new}} \leftarrow \dfrac{d + \sqrt{c} \cdot V_f\left(d/\sqrt{c}, \varepsilon\sqrt{c}\right)}{1 - W_f\left(d/\sqrt{c}, \varepsilon\sqrt{c}\right)}$ <br><br> $c := \pi_x - \pi_{f \to x}, \qquad d := \tau_x - \tau_{f \to x}$ |

Table 1: The update equations for the (cached) marginals $p(x)$ and the messages $m_{f \to x}$ for all factor types of a TrueSkill factor graph. We represent Gaussians $\mathcal{N}(\cdot; \mu, \sigma)$ in terms of their canonical parameters: precision, $\pi := \sigma^{-2}$, and precision adjusted mean, $\tau := \pi\mu$. The missing update equation for the message or the marginal follow from (6).

moment matching is known to minimise the Kullback–Leibler divergence. Then, we exploit the fact that from (3) and (5) we have

$$\hat{p}(d_i) = \hat{m}_{f \to d_i}(d_i) \cdot m_{d_i \to f}(d_i) \qquad \Leftrightarrow \qquad \hat{m}_{f \to d_i}(d_i) = \frac{\hat{p}(d_i)}{m_{d_i \to f}(d_i)}. \qquad (6)$$

Table 1 gives all the update equations necessary for performing inference in the TrueSkill factor graph. The top four rows result from standard Gaussian integrals. The bottom rule is the result of the moment matching procedure described above. The four functions are the additive and multiplicative correction term for the mean and variance of a (doubly) truncated Gaussian and are given by (see also Figure 2):

$$V_{\mathbb{I}(\cdot > \varepsilon)}(t, \varepsilon) := \frac{\mathcal{N}(t - \varepsilon)}{\Phi(t - \varepsilon)}, \quad W_{\mathbb{I}(\cdot > \varepsilon)}(t, \varepsilon) := V_{\mathbb{I}(\cdot > \varepsilon)}(t, \varepsilon) \cdot \left(V_{\mathbb{I}(\cdot > \varepsilon)}(t, \epsilon) + t - \varepsilon\right),$$

$$V_{\mathbb{I}(|\cdot|>\varepsilon)}(t,\varepsilon) \quad := \quad \frac{\mathcal{N}(-\varepsilon - t) - \mathcal{N}(\varepsilon - t)}{\Phi(\varepsilon - t) - \Phi(-\varepsilon - t)},$$

$$W_{\mathbb{I}(|\cdot|>\varepsilon)}(t,\varepsilon) \quad := \quad V^2_{\mathbb{I}(|\cdot|>\varepsilon)}(t,\varepsilon) + \frac{(\varepsilon - t) \cdot \mathcal{N}(\varepsilon - t) + (\varepsilon + t)\,\mathcal{N}(\varepsilon + t)}{\Phi(\varepsilon - t) - \Phi(-\varepsilon - t)}.$$

Since the messages 2 and 5 are approximate, we need to iterate over all messages that are on the shortest path between any two approximate marginals $\hat{p}(d_i)$ until the approximate marginals do not change anymore. The resulting optimal message passing schedule can be found in Figure 1 (arrows and caption).

# 4 Experiments and Online Service

## 4.1 Halo 2 Beta Test

In order to assess the performance of the TrueSkill algorithm we performed experiments on the game outcome data set generated by Bungie Studios during the beta testing of the Xbox title Halo 2[2]. The data set consists of thousands of game outcomes for four different types of games: 8 players against each other ("Free for All"), 4 players vs. 4 players ("Small Teams"), 1 player vs. 1 player ("Head to Head"), and 8 players vs. 8 players ("Large Teams"). The draw margin $\varepsilon$ for each factor node was set by counting the fraction of draws between teams ("empirical draw probability") and relate the draw margin $\varepsilon$ to the chance of drawing by

$$\mathrm{draw\,probability} = \Phi\left(\frac{\varepsilon}{\sqrt{n_1 + n_2}\beta}\right) - \Phi\left(\frac{-\varepsilon}{\sqrt{n_1 + n_2}\beta}\right) = 2\Phi\left(\frac{\varepsilon}{\sqrt{n_1 + n_2}\beta}\right) - 1\,,$$

where $n_1$ and $n_2$ are the number of players in each of the two teams compared by a $\mathbb{I}(\cdot > \varepsilon)$ or $\mathbb{I}(|\cdot| \le \varepsilon)$ node (see Figure 1). The performance variance $\beta^2$ and the dynamics variance $\gamma^2$ were set to the standard values (see next section). We compared the TrueSkill algorithm to Elo with a Gaussian performance distribution (1) and $\alpha = 0.07$; this corresponds to a $K$ factor of 24 on the Elo scale which is considered a good and stable dynamics (see [4]). When we had to process a team game or a game with more than two teams we used the so-called *duelling* heuristic: For each player, compute the $\Delta$'s in comparison to all other players based on the team outcome of the player and every other player and perform an update with the average of the $\Delta$'s. The approximate message passing algorithm described in the last section is extremely efficient; in all our experiments the runtime of the ranking algorithm was within twice the runtime of the simple Elo update.

**Predictive Performance** The following table presents the prediction error (fraction of teams that were predicted in the wrong order before the game) for both algorithms (column 2 and 3). This measure is difficult to interpret because of the interplay of ranking and matchmaking: Depending on the (unknown) true skills of all players, the smallest achievable prediction error could be as big as 50%. In order to compensate for this latent, unknown variable, we arranged a competition between ELO and TrueSkill: We let each system predict which games it considered most tightly matched and presented them to the other algorithm. The algorithm that predicts more game outcomes correctly has a better ability to identify tight matches. For TrueSkill we used the matchmaking criterion (7) and for Elo we used the difference in Elo scores, $s_1 - s_2$.

|  | ELO full | TrueSkill full | ELO "challenged" | TrueSkill "challenged" |
|---|---|---|---|---|
| Free for All | 32.14% | **30.82%** | 38.30% | **35.64%** |
| Small Teams | **34.92%** | 35.23% | 42.55% | **37.17%** |
| Head to Head | 33.24% | **32.44%** | 40.57% | **30.83%** |
| Large Teams | 39.49% | **38.15%** | 44.12% | **29.94%** |

It can be seen from column 4 and 5 of this table that TrueSkill is significantly better at predicting the tight matches (the "challenge" set was always 20% of the total number of games in each game mode).

## Match Quality

One of the main applications of a rating system is to be able to match players of similar skill. In order to compare the ability of Elo and TrueSkill on this task, we sorted the games based on the match quality assigned by both systems to each game. If the match was truly tight then it would be very likely to observe a draw. Thus, we plot the fraction of draws (out of all possible draws) accumulating over the match quality order assigned by each system. In the graph on the right we see that TrueSkill is significantly better than Elo for both the "Free for All" and "Head to Head" game mode but fails in "Small Teams". This is possibly due to the violation of the additive team performance model as most games in this mode are Capture-the-Flag games.

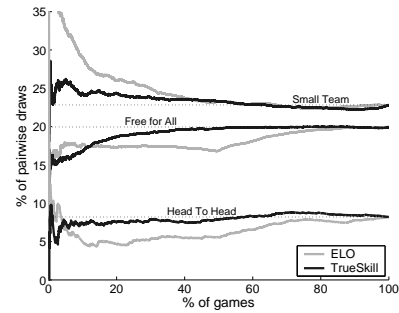

## Win Probability

The perceived quality of a rating system for players is in terms of their winning ratio: if the winning ratio is high then player was erroneously assigned too weak opposition by the ranking system (and vice versa). In a second experiment we processed the Halo 2 dataset but rejected games that did not meet a certain match quality threshold. For the games thus selected, we computed the winning ratio of each player and, depending on the minimal number of games played by each player, measured the average deviation of the winning probability from 50% (the "optimal" winning ratio). The resulting plot on the right (for the "Head to Head" game mode) shows that with TrueSkill ever players with very few games got mostly fair matches (with a winning probability within 35% to 65%).

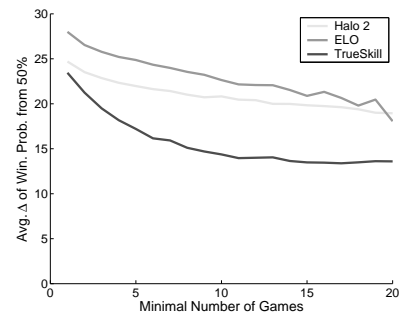

## Convergence Properties

Finally, we plotted two exemplary convergence trajectories for two of the highest rated players in the "Free for All" game mode (Solid line: TrueSkill; Dashed line: Elo). As can be seen, TrueSkill automatically chooses the correct learning rate whereas Elo only slowly converges to the target skill. In fact, TrueSkill comes close to the information theoretic limit of $n \log(n)$ bits to encode a ranking of $n$ players. For 8 player games, the information theoretic limit is $\log(n)/\log(8) \approx 5$ games per player on average and the observed convergence for these two players is $\approx 10$ games!

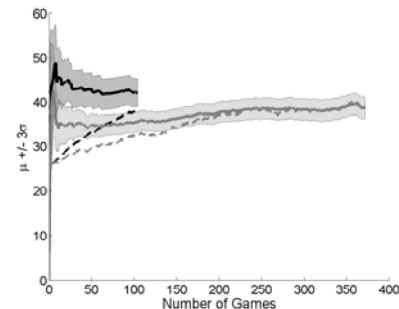

## 4.2  TrueSkill in Xbox 360 Live

Xbox Live is Microsoft's console online gaming service. It lets players play together across the world in hundreds of different titles. As of September 2005 Xbox Live had over 2 million subscribed users who had accrued over 1.3 billion hours on the service. The new and improved Xbox 360 Live service offers automatic player rating and matchmaking using the TrueSkill algorithm. The system processes hundreds of thousands of games per day making it one of the largest applications of Bayesian inference to date.

In Xbox Live we use a scale given by a prior $\mu_0 = 25$ and $\sigma_0^2 = (25/3)^2$ corresponding to a probability for positive skills of approximately 99%. The variance of performance is given

by $\beta^2 = (\sigma_0/2)^2$ and the dynamics variance is chosen to be $\gamma^2 = (\sigma_0/100)^2$. The TrueSkill skill of a player $i$ is currently displayed as a conservative skill estimate given by the 1% lower quantile $\mu_i - 3\sigma_i$. This choice ensures that the top of the leaderboards (a listing of all players according to $\mu - 3\sigma$) are only populated by players that are highly skilled with high certainty, having worked up their way from $0 = \mu_0 - 3\sigma_0$. Pairwise matchmaking of players is performed using a match quality criterion derived as the draw probability relative to the highest possible draw probability in the limit $\varepsilon \to 0$,

$$q_{\text{draw}}\left(\beta^2, \mu_i, \mu_j, \sigma_i, \sigma_j\right) := \sqrt{\frac{2\beta^2}{2\beta^2 + \sigma_i^2 + \sigma_j^2}} \cdot \exp\left(-\frac{(\mu_i - \mu_j)^2}{2\left(2\beta^2 + \sigma_i^2 + \sigma_j^2\right)}\right) . \qquad (7)$$

Note that the matchmaking process can be viewed as a process of sequential experimental design [3]. Since the quality of a match is determined by the unpredictability of its outcome, the goals of matchmaking and finding the most informative matches are aligned!

As a fascinating by-product we have the opportunity to study TrueSkill in action with player populations of hundreds of thousands of players. While we are only just beginning to analyse the vast amount of resulting data, we have already made some interesting observations.

1. Games differ in the number of effective skill levels. Games of chance (e.g., single game Backgammon or UNO) have a narrow skill distribution while games of skill (e.g., semi-realistic racing games) have a wide skill distribution.
2. Matchmaking and skill display result in a feedback loop back to the players, who often view their skill estimate as a reward or punishment for performance. Some players try to protect or boost their skill rating by either stopping to play, by carefully choosing their opponents, or by cheating.
3. The total skill distribution is shifted to below the prior distribution if players new to the system consistently lose their first few games. When a skill reset was initiated, we found that the effect disappeared with tighter matchmaking enforced.

## 5 Conclusion

TrueSkill is a globally deployed Bayesian skill rating algorithm based on approximate message passing in factor graphs. It has many theoretical and practical advantages over the Elo system and has been demonstrated to work well in practice.

While we specifically focused on the TrueSkill algorithm, many more interesting models can be developed within the factor graph framework presented here. In particular, the factor graph formulation is applicable to the family of constraint classification models [6] that encompass a wide range of multiclass and ranking problems. Also, instead of ranking individual entities one can use feature vectors to build a ranking function, e.g., for web pages represented as bags-of-words. Finally, we are planning to run a full time-independent EP analysis across chess games to obtain TrueSkill ratings for chess masters of all times.

**Acknowledgements** We would like to thank Patrick O'Kelley, David Shaw and Chris Butcher for interesting discussions. We also thank Bungie Studios for providing the data.

## Footnotes

[1] The transitive relation "1 draws with 2" is not modelled exactly by the relation $|t_1 - t_2| \leq \varepsilon$, which is non-transitive. If $|t_1 - t_2| \leq \varepsilon$ and $|t_2 - t_3| \leq \varepsilon$ then the model generates a draw among the three teams despite the possibility that $|t_1 - t_3| > \varepsilon$.

[2]Available for download at `http://research.microsoft.com/mlp/apg/downloads.htm`

## References

[1] A. H. David. *The Method of Paired Comparisons*. Charles Griffin and Company, London, 1969.

[2] A. E. Elo. *The rating of chess players: Past and present*. Arco Publishing, New York, 1978.

[3] V. V. Fedorov. *Theory of optimal experiments*. Academic Press, New York, 1972.

[4] M. E. Glickman. A comprehensive guide to chess ratings. *Amer. Chess Journal*, 3:59–102, 1995.

[5] M. E. Glickman. Parameter estimation in large dynamic paired comparison experiments. *Applied Statistics*, 48:377–394, 1999.

[6] S. Har-Peled, D. Roth, and D. Zimak. Constraint classification: A new approach to multiclass classification and ranking. In *NIPS 15*, pages 785–792, 2002.

[7] F. R. Kschischang, B. Frey, and H.-A. Loeliger. Factor graphs and the sum-product algorithm. *IEEE Trans. Inform. Theory*, 47(2):498–519, 2001.

[8] T. Minka. *A family of algorithms for approximate Bayesian inference*. PhD thesis, MIT, 2001.